# Discriminant Saliency for Visual Recognition from Cluttered Scenes

**Dashan Gao**          **Nuno Vasconcelos**
Department of Electrical and Computer Engineering,
University of California, San Diego

## Abstract

Saliency mechanisms play an important role when visual recognition must be performed in cluttered scenes. We propose a computational definition of saliency that deviates from existing models by equating saliency to discrimination. In particular, the salient attributes of a given visual class are defined as the features that enable best discrimination between that class and all other classes of recognition interest. It is shown that this definition leads to saliency algorithms of low complexity, that are scalable to large recognition problems, and is compatible with existing models of early biological vision. Experimental results demonstrating success in the context of challenging recognition problems are also presented.

## 1   Introduction

The formulation of recognition as a problem of statistical classification has enabled significant progress in the area, over the last decades. In fact, for certain types of problems (face detection/recognition, vehicle detection, pedestrian detection, etc.) it now appears to be possible to design classifiers that "work reasonably well most of the time", i.e. classifiers that achieve high recognition rates in the absence of a few factors that stress their robustness (e.g. large geometric transformations, severe variations of lighting, etc.). Recent advances have also shown that real-time recognition is possible on low-end hardware [1]. Given all this progress, it appears that one of the fundamental barriers remaining in the path to a vision of scalable recognition systems, capable of dealing with large numbers of visual classes, is an issue that has not traditionally been considered as problematic: training complexity. In this context, an aspect of particular concern is the dependence, of most modern classifiers, on carefully assembled and pre-processed training sets. Typically these training sets are large (in the order of thousands of examples per class) and require a combination of 1) painstaking manual labor of image inspection and segmentation of good examples (e.g. faces) and 2) an iterative process where an initial classifier is applied to a large dataset of unlabeled data, the classification results are manually inspected to detect more good examples (usually examples close to the classification boundary, or where the classifier fails), and these good examples are then manually segmented and added to the training set.

Overall, the process is extremely laborious, and good training sets usually take years to establish through the collaborative efforts of various research groups. This is completely opposite to what happens in truly scalable learning systems (namely biological ones) that are able to quickly bootstrap the learning process from a small number of virtually unprocessed examples. For example while humans can bootstrap learning with weak clues and highly cluttered scenes (such as "Mr. X is the person sitting at the end of the room, the one with gray hair"), current faces detectors require training faces to be cropped into

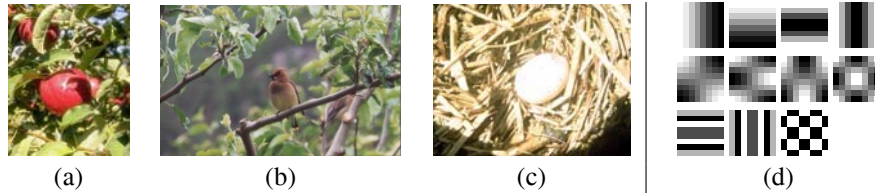

|     |     |     |     |
| :-: | :-: | :-: | :-: |
| (a) | (b) | (c) | (d) |

Figure 1: (a)(b)(c) Various challenging examples for current saliency detectors. (a) Apple hanging from a tree; (b) a bird in a tree; (c) an egg in a nest. (d) some DCT basis functions. From left to right, top to bottom, detectors of: edges, bars, corners, t-junctions, spots, flow patches, and checkerbords.

$20 \times 20$ pixel arrays, with all the hair precisely cropped out, lighting gradients explicitly removed, and so on. One property of biological vision that plays an important role in this ability to learn from highly cluttered examples, is the existence of saliency mechanisms. For example, humans rarely have to exhaustively scan a scene to detect an object of interest. Instead, salient locations simply pop-out in result of the operation of pre-recognition saliency mechanisms. While saliency has been the subject of significant study in computer vision, most formulations do not pose saliency itself as a major goal of recognition. Instead saliency is usually an auxiliary pre-filtering step, whose goal is to reduce computation by eliminating image locations that can be universally classified as non-salient, according to some definition of saliency which is completely divorced from the particular recognition problem at hand.

In this work, we propose an alternative definition of saliency, which we denote by *discriminant saliency*, and which is intrinsically grounded on the recognition problem. This new formulation is based on the intuition that, for recognition, the salient features of a visual class are those that best distinguish it from all other visual classes of recognition interest. We show that 1) this intuition translates naturally into a computational principle for the design of saliency detectors, 2) this principle can be implemented with great computational simplicity, 3) it is possible to derive implementations which scale to recognition problems with large numbers of classes, and 4) the resulting saliency mechanisms are compatible with classical models of biological perception. We present experimental results demonstrating success on image databases containing complex scenes and substantial amounts of clutter.

## 2 Saliency detection

The extraction of salient points from images has been a subject of research for at least a few decades. Broadly speaking, saliency detectors can be divided into three major classes. The first, and most popular, treats the problem as one of the *detection of specific visual attributes*. These are usually edges or corners (also called "interest points") [2] and their detection has roots in the structure-from-motion literature, but there have also been proposals for other low-level visual attributes such as contours [3]. A major limitation of these saliency detectors is that they do not generalize well. For example, a corner detector will always produce a stronger response in a region that is strongly textured than in a smooth region, even though textured surfaces are not necessarily more salient than smooth ones. This is illustrated by the image of Figure 1(a). While a corner detector would respond strongly to the highly textured regions of leaves and tree branches, it is not clear that these are more salient than the smooth apple. We would argue for the contrary.

Some of these limitations are addressed by more recent, and generic, formulations of saliency. One idea that has recently gained some popularity is to define *saliency as image complexity*. Various complexity measures have been proposed in this context. Lowe [4] measures complexity by computing the intensity variation in an image using the difference of Gaussian function; Sebe [5] measures the absolute value of the coefficients of a wavelet decomposition of the image; and Kadir [6] relies on the entropy of the distribution of local

intensities. The main advantage of these *data-driven* definitions of saliency is a significantly greater flexibility, as they could detect any of the low-level attributes above (corners, contours, smooth edges, etc.) depending on the image under consideration. It is not clear, however, that saliency can always be equated with complexity. For example, Figures 1(b) and (c), show images containing complex regions, consisting of clustered leaves and straw, that are not terribly salient. On the contrary, the much less complex image regions containing the bird or the egg appear to be significantly more salient.

Finally, a third formulation is to start from *models of biological vision*, and derive saliency detection algorithms from these models [7]. This formulation has the appeal of its roots on what are the only known full-functioning vision systems, and it has been shown to lead to interesting saliency behavior [7]. However, these solutions have one significant limitation: the lack of a clearly stated optimality criteria for saliency. In the absence of such a criteria it is difficult to evaluate, in an objective sense, the goodness of the proposed algorithms or to develop a theory (and algorithms) for optimal saliency.

## 3    Discriminant saliency

The basic intuition for discriminant saliency is somewhat of a "statement of the obvious": *the salient attributes of a given visual concept are the attributes that most distinguish it from all other visual concepts that may be of possible interest*. While close to obvious, this definition is a major departure from all existing definitions in the vision literature.

First, it makes reference to a "set of visual concepts of possible interest". While such a set may not be well defined for all vision problem (e.g. tracking or structure-from-motion where many of the current saliency detectors have roots [2]), it is an intrinsic component of the recognition problem: the set of visual classes to be recognized. It therefore makes saliency contingent upon the existence of a collection of classes and, therefore, impossible to compute from an isolated image. It also means that, for a given object, different visual attributes will be salient in different recognition contexts. For example while contours and shape will be most salient to distinguish a red apple from a red car, color and texture will be most salient when the same apple is compared to an orange. All these properties appear to be a good idea for recognition. Second, it sets as a goal for saliency that of distinguishing between classes. This implies that the optimality criterion for the design of salient features is discrimination, and therefore very different from traditional criteria such as repetitiveness under image transformations [8]. Robustness in terms of these criteria (which, once again, are well justified for tracking but do not address the essence of the recognition problem) can be learned if needed to achieve discriminant goals [9].

Due to this equivalence between saliency and discrimination, the principle of discriminant saliency can be easily translated into an optimality criteria for the design of saliency algorithms. In particular, it is naturally formulated as an optimal feature selection problem: optimal features for saliency are the most discriminant features for the one-vs-all classification problem that opposes the class of interest to all remaining classes. Or, in other words, the most salient features are the ones that best separate the class of interest from all others. Given the well known equivalence between features and image filters, this can also be seen as a problem of designing optimal filters for discrimination.

### 3.1    Scalable feature selection

In the context of scalable recognition systems, the implementation of discriminant saliency requires 1) the design of a large number of classifiers (as many as the total number of classes to recognize) at set up time, and 2) classifier tuning whenever new classes are added to, or deleted from, the problem. It is therefore important to adopt feature selection techniques that are computationally efficient, preferably reusing computation from the design of one classifier to the next. The design of such feature selection methods is a non-trivial problem, which we have been actively pursuing in the context of research in feature selection itself [11]. This research has shown that information-theoretic methods,

based on maximization of mutual information between features and class labels, have the appealing property of enabling a precise control (through factorizations based on known statistical properties of images) over the trade off between optimality, in a minimum Bayes error sense, and computationally efficiency [11]. Our experience of applying algorithms in this family to the saliency detection problem is that, even those strongly biased towards efficiency can consistently select good saliency detection filters. This is illustrated by all the results presented in this paper, where we have adopted the maximization of marginal diversity (MMD) [10] as the guiding principle for feature selection.

Given a classification problem with class labels $Y$, prior class probabilities $P_Y(i)$, a set of $n$ features, $\mathbf{X} = (X_1, \ldots, X_n)$, and such that the probability density of $X_k$ given class $i$ is $P_{X_k|Y}(x|i)$, the marginal diversity (MD) of feature $X_k$ is

$$\mathbf{md}(X_k) = < KL[P_{X_k|Y}(x|i)||P_{X_k}(x) >_Y \tag{1}$$

where $< f(i) >_Y = \sum_{i=1}^{M} P_Y(i)f(i)$, and $KL[p||q] = \int p(s) \log \frac{p(x)}{q(x)} dx$ the Kullback-Leibler divergence between p and q. Since it only requires marginal density estimates, the MD can be computed with histogram-based density estimates leading to an extremely efficient algorithm for feature selection [10]. Furthermore, in the one-vs-all classification scenario, the histogram of the "all" class can be obtained by a weighted average of the class conditional histograms of the image classes that it contains, i.e.

$$P_{X_k|Y}(x|\mathcal{A}) = \sum_{i \in \mathcal{A}} P_{X_k|Y}(x|i)P_Y(i) \tag{2}$$

where $\mathcal{A}$ is the set of image classes that compose the "all" class. This implies that the bulk of the computation, the density estimation step, only has to be performed once for the design of all saliency detectors.

### 3.2 Biologically plausible models

Ever since Hubel and Wiesel's showing that different groups in V1 are tuned for detecting different types of stimulae (e.g. bars, edges, etc.) it has been known that, the earliest stages of biological vision can be modeled as a multiresolution image decomposition followed by some type of non-linearity. Indeed, various "biologically plausible" models of early vision are based on this principle [12]. The equivalence between saliency detection and the design of optimally discriminant filters, makes discriminant saliency compatible with most of these models. In fact, as detailed in the experimental section, our experience is that remarkably simple mechanisms, inspired by biological vision, are sufficient to achieve good saliency results. In particular, all the results reported in this paper were achieved with the following two step procedure, based on the Malik-Perona model of texture perception [13]. First, a saliency map (i.e. a function describing the saliency at each pixel location) is obtained by pooling the responses of the different saliency filters after half-wave rectification

$$S(x,y) = \sum_{i=1}^{2n} \omega_i R_i^2(x,y), \tag{3}$$

where $S(x,y)$ is the saliency at location $(x,y)$, $R_i(x,y), i = 1, \ldots, 2n$ the channels resulting from half-wave rectification of the outputs of the saliency filters $F_i(x,y), i = 1, \ldots, n$

$$R_{2k-1} = \max[-I * F_k(x,y), 0] \qquad\qquad R_{2k} = \max[I * F_k(x,y), 0] \tag{4}$$

$I(x,y)$ the input image, and $w_i = md(i)$ a weight equal to the feature's marginal diversity. Second, the saliency map is fed to a peak detection module that consists of a winner-take-all network. The location of largest saliency is first found. Its spatial scale is set to the size of the region of support of the saliency filter with strongest response at that location. All neighbors within a circle whose radius is this scale are then suppressed (set to zero) and the process is iterated. The procedure is illustrated by Figure 2, and produces

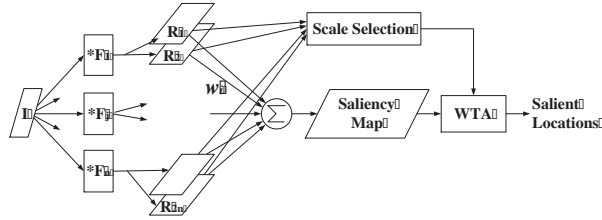

Figure 2: Schematic of the saliency detection model.

a list of salient locations, their saliency strengths, and scales. It is important to limit the number of channels that contribute to the saliency map since, for any given class, there are usually many features which are not discriminant but have strong response at various image locations (e.g. areas of clutter). This is done through a cross-validation step that we discuss in section 4.3.

All the experiments presented in the following section were obtained using the coefficients of the discrete cosine transform (DCT) as features. While the precise set of features is likely not to be crucial for the quality of the saliency results (e.g. other invertible multiresolution decompositions, such as Gabor or other wavelets, would likely work well) the DCT feature set has two appealing properties. First, previous experience has shown that they perform quite well in large scale recognition problems [14]. Second, as illustrated by Figure 1(d), the DCT basis functions contain detectors for various perceptually relevant low-level image attributes, including edges, bars, corners, t-junctions, spots, etc. This can obviously only make the saliency detection process easier.

## 4 Results and discussion

We start the experimental evaluation of discriminant saliency with some results from the Brodatz texture database, that illustrate various interesting properties of the former.

### 4.1 Saliency maps

Brodatz is an interesting database because it stresses aspects of saliency that are quite problematic for most existing saliency detection algorithms: 1) the need to perform saliency judgments in highly textured regions, 2) classes that contain salient regions of diverse shapes, and 3) a great variety of salient attributes - e.g. corners, closed and open contours, regular geometric geometric figures (circles, squares, etc.), texture gradients, crisp and soft edges, etc. The entire collection of textures in the database was divided into a train and test set, using the set-up of our previous retrieval work [14]. The training database was used to determine the salient features of each class, and saliency maps were then computed on the test images. The process was repeated for all texture classes, on a one-vs-all setting (class of interest against all others) with each class sequentially considered as the "one" class.

As illustrated by the examples shown in Figure 3, none of the challenges posed by Brodatz seem very problematic for discriminant saliency. Note, in particular, that the latter does not appear to have any difficulty in 1) ignoring highly textured background areas in favor of a more salient foreground object (two leftmost images), which could itself be another texture, 2) detecting as salient a wide variety of shapes, contours of different crispness and scale, or 3) even assign strong saliency to texture gradients (rightmost image). This robustness is a consequence of the fact that the saliency features are tuned to discriminate the class of interest from the rest. We next show that it can lead to significantly better saliency detection performance than that achievable with the algorithms currently available in the literature.

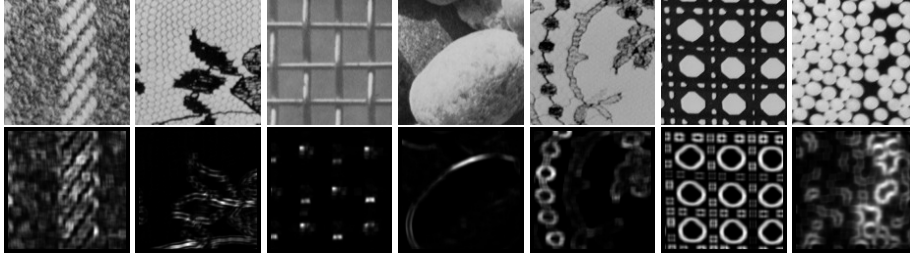

Figure 3: Saliency maps (bottom row) obtained on various textures (shown in top row) from Brodatz. Bright pixels flag salient locations. Note: the saliency maps of the second row are best viewed on paper. A gamma-corrected version would be best for viewing on CRT displays and is available at www.svcl.ucsd.edu/publications/nips04-crt.ps

| Dataset | DSD | SSD | HSD | pixel-based | constellation [15] |
|---------|-----|-----|-----|-------------|--------------------|
| Faces | 97.24 | 77.3 | 61.87 | 93.05 | 96.4 |
| Motorbikes | 96.25 | 81.3 | 74.83 | 87.83 | 92.5 |
| Airplanes | 93.00 | 78.7 | 80.17 | 90.33 | 90.2 |

Table 1: SVM classification accuracy based on different detectors.

## 4.2 Comparison to existing methods

While the results of the previous section provide interesting anecdotal evidence in support of discriminant saliency, objective conclusions can only be drawn by comparison to existing techniques. Unfortunately, it is not always straightforward to classify saliency detectors objectively by simple inspection of saliency maps, since different people frequently attribute different degrees of saliency to a given image region. In fact, in the absence of a larger objective for saliency, e.g. recognition, it is not even clear that the problem is well defined. To avoid the obvious biases inherent to a subjective evaluation of saliency maps, we tried to design an experiment that could lead to an objective comparison. The goal was to quantify if the saliency maps produced by the different techniques contained enough information for recognition. The rational is the following. If, when applied to an image, a saliency detector has an output which is highly correlated with the appearance/absence of the class of interest in that image, then it should be possible to classify the image (as belonging/not belonging to the class) by classifying the saliency map itself. We then built the simplest possible saliency map classifier that we could conceive of: the intensity values of the saliency map were histogrammed and fed to a support vector machine (SVM) classifier.

We compared the performance of the discriminant saliency detector (DSD) described above, with one representative from each of the areas of the literature discussed in section 2: the Harris saliency detector (HSD) and the scale saliency detector (SSD) of [6]. To evaluate performance on a generic recognition scenario, we adopted the Caltech database, using the experimental set up proposed in [15]. To obtain an idea of what would be acceptable classification results on this database, we used two benchmarks: the performance, on the same classification task, of 1) a classifier of equivalent simplicity but applied to the images themselves and 2) the constellation-based classifier proposed in [15] (which we believe to be representative of the state-of-the-art for this database). For the simple classifier, we reduced the luminance component of each image to a vector (by stacking all pixels into a column) and used a SVM to classify the resulting set of points. All parameters were set to assure a fair comparison between the saliency detectors (e.g. a multiscale version of Harris was employed, all detectors combined information from three scales, etc.). Table 1 presents the two benchmarks and the results of classifying the saliency histograms generated by the three detectors.

The table supports various interesting conclusions. First, both the HSD and the SSD have

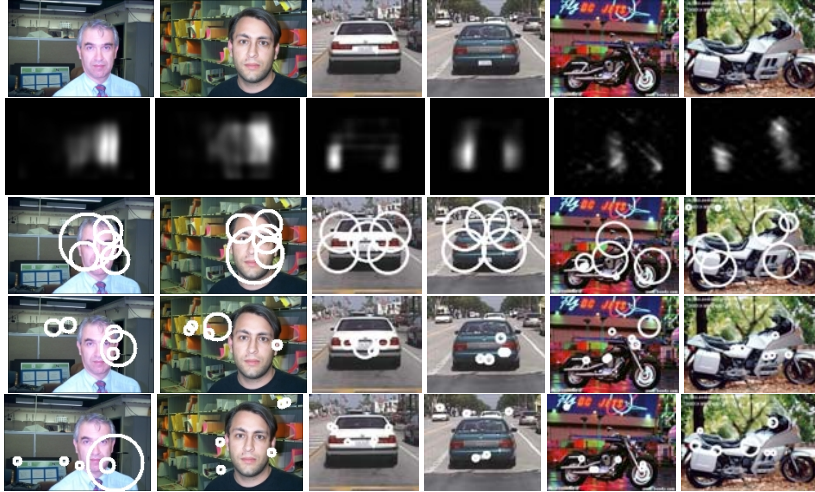

Figure 4: Original images (top row), saliency maps generated by DSD (second row), and a comparison of salient locations detected by: DSD in the third row, SSD in the fourth, and HSD at the bottom. Salient locations are the centers of the white circles, the circle radii representing scale. Note: the saliency maps of the second row are best viewed on paper. A gamma-corrected version would be best for viewing in CRT displays and is available at www.svcl.ucsd.edu/svclwww/publications/nips04-crt.ps

very poor performance, indicating that they produce saliency maps that have weak correlation with the presence/absence of the class of interest in the image to classify. Second, the simple pixel-based classifier works surprisingly well on this database, given that there is indeed a substantial amount of clutter in its images (see Figure 4). Its performance is, nevertheless, inferior to that of the constellation classifier. The third, and likely most surprising, observation is that the classification of the DSD histograms clearly outperform this classifier, achieving the overall best performance. It should be noted that this is somewhat of an unfair comparison for the constellation classifier, since it tries to solve a problem that is more difficult than the one considered in this experiment. While the question of interest here is "is class x present in the image or not?" this classifier can actually determine the location of the element from the class (e.g. a face) in the image. In any case, these results seem to support the claim that DSD produces saliency maps which contain most of the saliency information required for classification. The issue of translating these saliency maps into a combined segmentation/recognition solution will be addressed in future research.

Finally, the superiority of the DSD over the other two saliency detectors considered in this experiment is also clearly supported by the inspection of the resulting salient locations. Some examples are presented in Figure 4.

### 4.3 Determining the number of salient features

In addition to experimental validation of the performance of discriminant saliency, the experiment of the previous section suggests a classification-optimal strategy to determine the number of features that contribute to the saliency maps of a given class of interest. Note that, while the training examples from each class are not carefully segmented (and can contain large areas of clutter), the working assumption is that each image is labeled with respect to the presence or absence in it of the class of interest. Hence, the classification problem of the previous section is perfectly well defined before segmentation (e.g. separation of the pixels containing objects in the class and pixels of background) takes place. It follows that a natural way to determine the optimal number of features is to search for the number that maximizes the classification rate on this problem. This search can be performed by

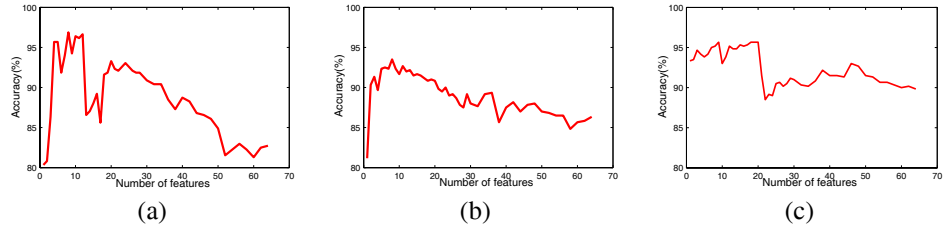

Figure 5: Classification accuracy vs number of features considered by the saliency detector for (a) faces, (b) motorbikes and (c) airplanes.

a traditional cross-validation strategy, the strategy that we have adopted for all the results presented in this paper. One interesting question is whether the performance of the DSD is very sensitive to the number of features chosen. Our experience is that, while it is important to limit the number of features, there is usually a range that leads to results very close to optimal. This is shown in Figure 5 where we present the variation of the classification rate on the problem of the previous section for various classes on Caltech. Visual inspection of the saliency detection results obtained with feature sets within this range showed no substantial differences with respect to that obtained with the optimal feature set.

## References

[1] P. Viola and M. Jones. Robust real-time object detection. $2^{nd}$ Int. Workshop on Statistical and Computational Theories of Vision Modeling, Learning, Computing and Sampling, July 2001.

[2] C. Harris and M. Stephens. A combined corner and edge detector. Alvey Vision Conference, 1988.

[3] A. Sha'ashua and S. Ullman. Structural saliency: the detection of globally salient structures using a locally connected network. Proc. Internat. Conf. on Computer Vision, 1988.

[4] D. G. Lowe. Object recognition from local scale-invariant features. In Proceedings of International Conference on Computer Vision, pp. 1150-1157, 1999.

[5] N. Sebe, M. S. Lew. Comparing salient point detectors. Pattern Recognition Letters, vol.24, no.1-3, Jan. 2003, pp.89-96.

[6] T. Kadir and M.l Brady. Scale, Saliency and Image Description. International Journal of Computer Vision, Vol.45, No.2, p83-105, November 2001

[7] L. Itti, C. Koch and E. Niebur. A model of saliency-based visual attention for rapid scene analysis. IEEE Trans. Pattern Analysis and Machine Intelligence, 20(11), Nov. 1998.

[8] C. Schmid, R. Mohr and C. Bauckhage. Comparing and Evaluating Interest Points. Proceedings of International Conference on Computer Vision 1998, p.230-235.

[9] D. Claus and A. Fitzgibbon. Reliable Fiducial Detection in Natural Scenes. Proceedings of the 8th European Conference on Computer Vision, Prague, Czech Republic, 2004

[10] N. Vasconcelos. Feature Selection by Maximum Marginal Diversity. In Neural Information Processing System, Vancouver, Canada, 2002

[11] N. Vasconcelos. Scalable Discriminant Feature Selection for Image Retrieval and Recgnition. To appear in Proc. of IEEE Conf. on Computer Vision and Pattern Recognition (CVPR), 2004

[12] D. Sagi, "The Psychophysics of Texture Segmentation, in Early Vision and Beyond, T. Papathomas, Ed., chapter 7. MIT Press, 1996.

[13] J. Malik, P. Perona. Preattentive texture discrimination with early vision mechanisms. J Opt Soc Am A. 7(5), 1990 May, p923-32.

[14] N. Vasconcelos and G. Carneiro. What is the Role of Independence for Visual Recgnition? In Proc. European Conference on Computer Vision, Copenhagen, Denmark, 2002.

[15] R. Fergus, P. Perona and A. Zisserman. Object Class Recognition by Unsupervised Scale-Invariant Learning. In Proc. IEEE Conf. on Computer Vision and Pattern Recognition 2003.
